# Experiments with Neural Networks for Real Time Implementation of Control

**P. K. Campbell, M. Dale, H. L. Ferrá and A. Kowalczyk**

Telstra Research Laboratories
770 Blackburn Road Clayton, Vic. 3168, Australia
{p.campbell, m.dale, h.ferra, a.kowalczyk}@trl.oz.au

## Abstract

This paper describes a neural network based controller for allocating capacity in a telecommunications network. This system was proposed in order to overcome a "real time" response constraint. Two basic architectures are evaluated: 1) a feedforward network-heuristic and; 2) a feedforward network-recurrent network. These architectures are compared against a linear programming (LP) optimiser as a benchmark. This LP optimiser was also used as a teacher to label the data samples for the feedforward neural network training algorithm. It is found that the systems are able to provide a traffic throughput of 99% and 95%, respectively, of the throughput obtained by the linear programming solution. Once trained, the neural network based solutions are found in a fraction of the time required by the LP optimiser.

## 1 Introduction

Among the many virtues of neural networks are their efficiency, in terms of both execution time and required memory for storing a structure, and their practical ability to approximate complex functions. A typical drawback is the usually "data hungry" training algorithm. However, if training data can be computer generated off line, then this problem may be overcome. In many applications the algorithm used to generate the solution may be impractical to implement in real time. In such cases a neural network substitute can become crucial for the feasibility of the project. This paper presents preliminary results for a non-linear optimization problem using a neural network. The application in question is that of capacity allocation in an optical communications network. The work in this area is continuing and so far we have only explored a few possibilities.

## 2 Application: Bandwidth Allocation in SDH Networks

Synchronous Digital Hierarchy (SDH) is a new standard for digital transmission over optical fibres [3] adopted for Australia and Europe equivalent to the SONET (Synchronous Optical NETwork) standard in North America. The architecture of the particular SDH network researched in this paper is shown in Figure 1 (a).

1)  Nodes at the periphery of the SDH network are switches that handle individual calls.

2)  Each switch concentrates traffic for another switch into a number of streams.

3)  Each stream is then transferred to a Digital Cross-Connect (DXC) for switching and transmission to its destination by allocating to it one of several alternative virtual paths.

The task at hand is the dynamic allocation of capacities to these virtual paths in order to maximize SDH network throughput.

This is a non-linear optimization task since the virtual path capacities and the constraints, i.e. the physical limit on capacity of links between DXC's, are quantized, and the objective function (Erlang blocking) depends in a highly non-linear fashion on the allocated capacities and demands. Such tasks can be solved 'optimally' with the use of classical linear programming techniques [5], but such an approach is time-consuming - for large SDH networks the task could even require hours to complete.

One of the major features of an SDH network is that it can be remotely reconfigured using software controls. Reconfiguration of the SDH network can become necessary when traffic demands vary, or when failures occur in the DXC's or the links connecting them. Reconfiguration in the case of failure must be extremely fast, with a need for restoration times under 60 ms [1].

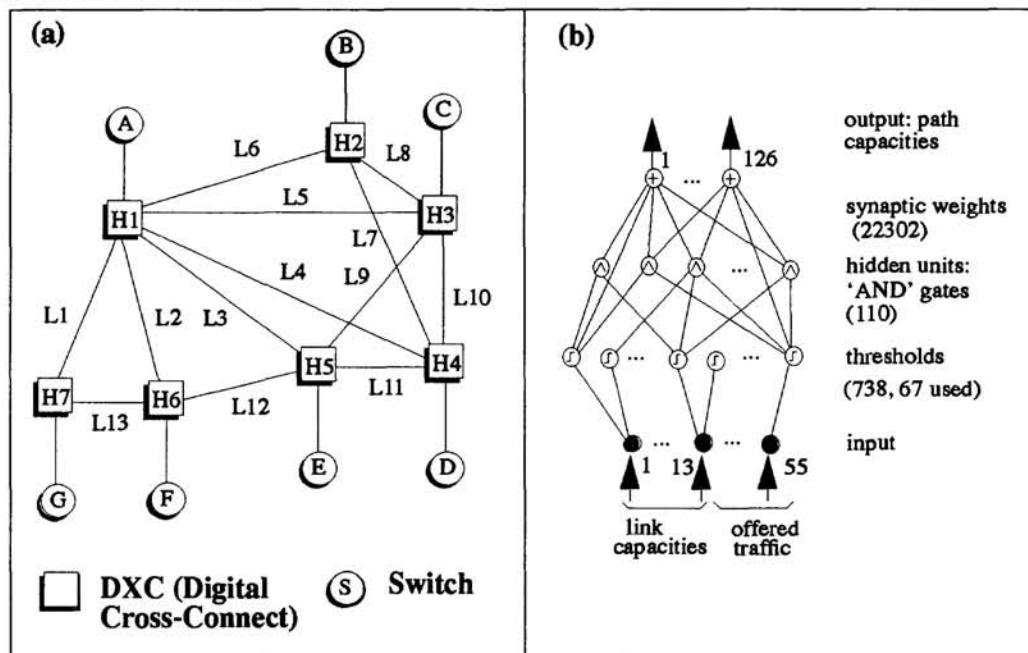

Figure 1
    (a) Example of an Inter-City SDH/SONET Network Topology used in experiments.
    (b) Example of an architecture of the mask perceptron generated in experiments.

In our particular case, there are three virtual paths allocated between any pair of switches, each using a different set of links between DXC's of the SDH network. Calls from one switch to another can be sent along any of the virtual paths, leading to 126 paths in total (7 switches to 6 other switches, each with 3 paths).

The path capacities are normally set to give a predefined throughput. This is known as the "steady state". If links in the SDH network become partially damaged or completely cut, the operation of the SDH network moves away from the steady state and the path capacities must be reconfigured to satisfy the traffic demands subject to the following constraints:

(i)   Capacities have integer values (between 0 and 64 with each unit corresponding to a 2 Mb/s stream, or 30 Erlangs),

(ii)  The total capacity of all virtual paths through any one link of the SDH network

cannot exceed the physical capacity of that link.

The neural network training data consisted of 13 link capacities and 42 traffic demand values, representing situations in which the operation of one or more links is degraded (completely or partially). The output data consisted of 126 integer values representing the difference between the steady state path capacities and the final allocated path capacities.

## 3 Previous Work

The problem of optimal SDH network reconfiguration has been researched already. In particular Gopal et. al. proposed a heuristic greedy search algorithm [4] to solve this non-linear integer programming problem. Herzberg in [5] reformulated this non-linear integer optimization problem as a linear programming (LP) task, Herzberg and Bye in [6] investigated application of a simplex algorithm to solve the LP problem, whilst Bye [2] considered an application of a Hopfield neural network for this task, and finally Leckie [8] used another set of AI inspired heuristics to solve the optimization task.

All of these approaches have practical deficiencies; the linear programming is slow, while the heuristic approaches are relatively inaccurate and the Hopfield neural network method (simulated on a serial computer) suffers from both problems.

In a previous paper Campbell et al. [10] investigated application of a mask perceptron to the problem of reconfiguration for a "toy" SDH network. The work presented here expands on the work in that paper, with the idea of using a second stage mask perceptron in a recurrent mode to reduce link violations/underutilizations.

## 4 The Neural Controller Architecture

Instead of using the neural network to solve the optimization task, e.g. as a substitute for the simplex algorithm, it is taught to replicate the optimal LP solution provided by it.

We decided to use a two stage approach in our experiments. For the first stage we developed a feedforward network able to produce an approximate solution. More precisely, we used a collection of 2000 random examples for which the linear programming solution of capacity allocations had been pre-computed to develop a feedforward neural network able to approximate these solutions.

Then, for a new example, such an "approximate" neural network solution was rounded to the nearest integer, to satisfy constraint (i), and used to seed the second stage providing refinement and enforcement of constraint (ii).

For the second stage experiments we initially used a heuristic module based on the Gopal et al. approach [4]. The heuristic firstly reduces the capacities assigned to all paths which cause a physical capacity violation on any links, then subsequently increases the capacities assigned to paths across links which are being under-utilized.

We also investigated an approach for the second stage which uses another feedforward neural network. The teaching signal for the second stage neural network is the difference between the outputs from the first stage neural network alone and the combined first stage neural network/heuristic solution. This time the input data consisted of 13 link usage values (either a link violation or underutilization) and 42 values representing the amount of traffic lost per path for the current capacity allocations. The second stage neural network had 126 outputs representing the correction to the first stage neural network's outputs.

The second stage neural network is run in a recurrent mode, adjusting by small steps the currently allocated link capacities, thereby attempting to iteratively move closer to the combined neural-heuristic solution by removing the link violations and under-utilizations left behind by the first stage network.

The setup used during simulation is shown in Figure 2. For each particular instance tested the network was initialised with the solution from the first stage neural network. The offered traffic (demand) and the available maximum link capacities were used to determine the extent of any link violations or underutilizations as well as the amount of lost traffic (demand satisfaction). This data formed the initial input to the second stage network. The outputs of the neural network were then used to check the quality of the

solution, and iteration continued until either no link violations occurred or a preset maximum number of iterations had been performed.

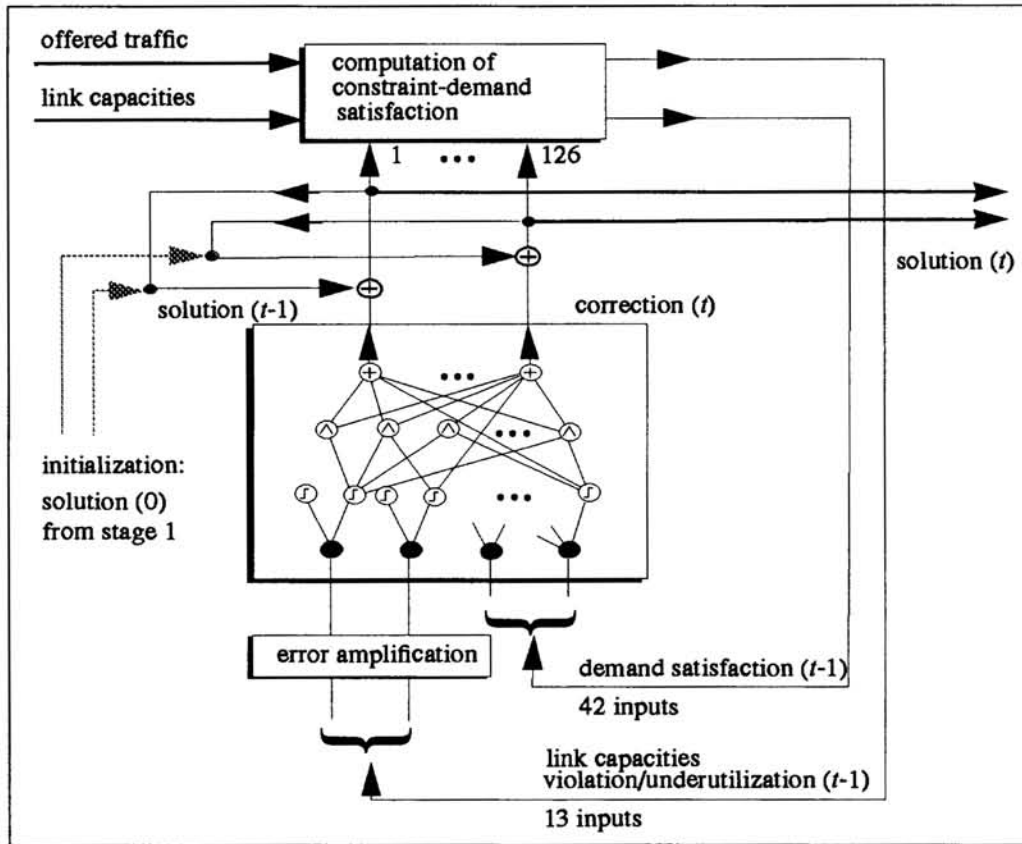

Figure 2. Recurrent Network used for second stage experiments.

When computing the constraint satisfaction the outputs of the neural network where combined and rounded to give integer link violations/under-utilizations. This means that in many cases small corrections made by the network are discarded and no further improvement is possible. In order to overcome this we introduced a scheme whereby errors (link violations/under-utilizations) are occasionally amplified to allow the network a chance of removing them. This scheme works as follows:

1) an instance is iterated until it has either no link violations or until 10 iterations have been performed;

2) if any link violations are still present then the size of the errors are multiplied by an amplification factor (>1);

3) a further maximum of 10 iterations are performed;

4) if subsequently link violations persist then the amplification factor is increased;

the procedure repeats until either all link violations are removed or the amplification factor reaches some fixed value.

## 5 Description of Neural Networks Generated

The first stage feedforward neural network is a mask perceptron [7], c.f. Figure 1 (b). Each input is passed through a number of arbitrarily chosen binary threshold units. There were a total of 738 thresholds for the 55 inputs. The task for the mask perceptron training algorithm [7] is to select a set of useful thresholds and hidden units out of thousands of possibilities and then to set weights to minimize the mean-square-error on the training set.

The mask perceptron training algorithm automatically selected 67 of these units for direct connection to the output units and a further 110 hidden units ("AND" gates) whose

outputs are again connected to the neural network outputs, giving 22,302 connections in all.

Such neural networks are very rapid to simulate since the only operations required are comparison and additions.

For the recurrent network used in the second stage we also used a mask perceptron. The training algorithm used for the recurrent network was the same as for the first stage, in particular note that no gradual adaptation was employed. The inputs to the network are passed through 589 arbitrarily chosen binary threshold units. Of these 35 were selected by the training algorithm for direct connection to the output units via 4410 weighted links.

## 6 Results

The results are presented in Table 1 and Figure 3. The values in the table represent the traffic throughput of the SDH network, for the respective methods, as a percentage of the throughput determined by the LP solution. Both the neural networks were trained using 2000 instances and tested against a different set of 2000 instances. However for the recurrent network approximately 20% of these cases still had link violations after simulation so the values in Table 1 are for the 80% of valid solutions obtained from either the training or test set.

| Solution type | Training | Test |
|---|---|---|
| Feedforward Net/Heuristic | 99.08% | 98.90%, |
| Feedforward Net/Recurrent Net | 94.93% (*) | 94.76%(*) |
| Gopal-S | 96.38% | 96.20% |
| Gopal-0 | 85.63% | 85.43% |

(*) these numbers are for the 1635 training and 1608 test instances (out of 2000) for which the recurrent network achieved a solution with no link violations after simulation as described in Section 3.

Table 1. Efficiency of solutions measured by average fraction of the 'optimal' throughput of the LP solution

As a comparison we implemented two solely heuristic algorithms. We refer to these as Gopal-S and Gopal-0. Both employ the same scheme described earlier for the Gopal et al. heuristic. The difference between the two is that Gopal-S uses the steady state solution as an initial starting point to determine virtual path capacities for a degraded network, whereas Gopal-0 starts from a point where all path capacities are initially set to zero.

Referring to Figure 3, *link capacity ratio* denotes the total link capacity of the degraded SDH network relative to the total link capacity of the steady state SDH network. A low value of link capacity ratio indicates a heavily degraded network. The *traffic throughput ratio* denotes the ratio between the throughput obtained by the method in question, and the throughput of the steady state solution.

Each dot in the graphs in Figure 3 represents one of the 2000 test set cases. It is clear from the figure that the neural network/heuristic approach is able to find better solutions for heavily degraded networks than each of the other approaches. Overall the clustering of dots for the neural network/heuristic combination is tighter (in the y-direction) and closer to 1.00 than for any of the other methods. The results for the recurrent network are very encouraging being qualitatively quite close to those for the Gopal-S algorithm.

All experiments were run on a SPARCStation 20. The neural network training took a few minutes. During simulation the neural network took an average of 9 ms per test case with a further 36.5 ms for the heuristic, for a total of 45.5 ms. On average the Gopal-S algorithm required 55.3 ms and the Gopal-0 algorithm required 43.7 ms per test case. The recurrent network solution required an average of 55.9 ms per test case. The optimal solutions calculated using the linear programming algorithm took between 2 and 60 seconds per case on a SPARCStation 10.

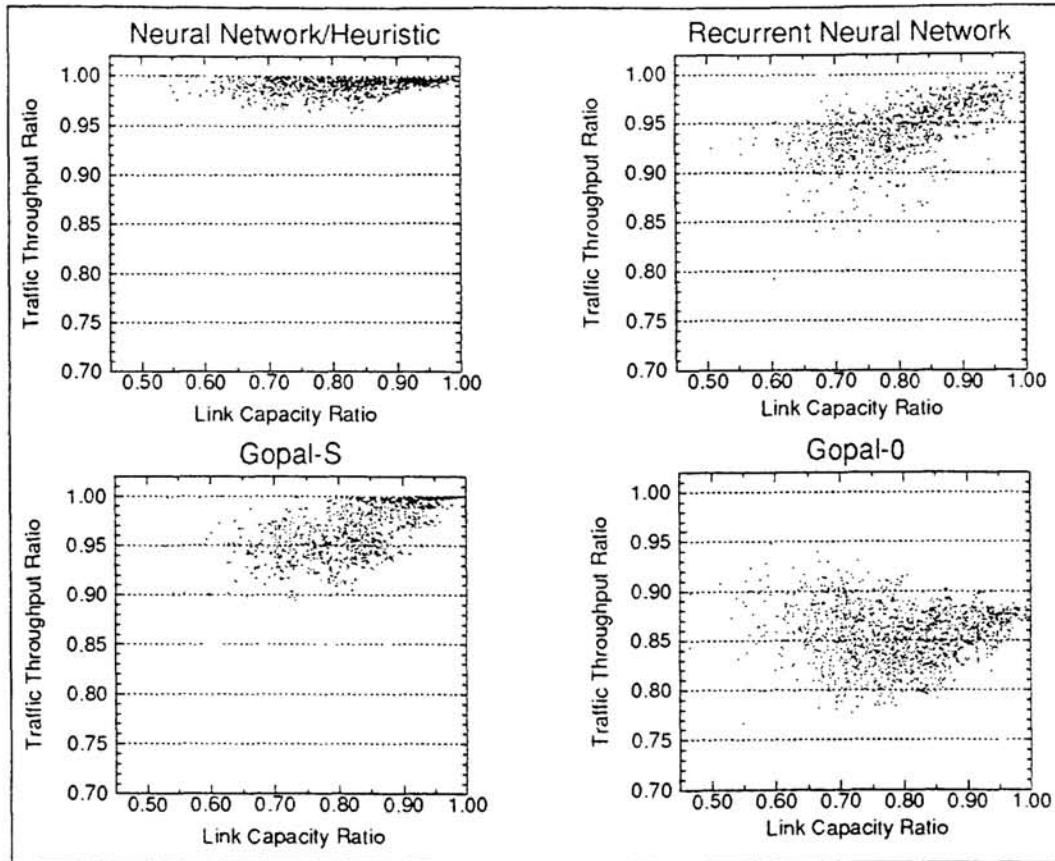

Figure 3. Experimental results for the Inter-City SDH network (Fig. 1) on the independent test set of 2000 random cases. On the x axis we have the ratio between the total link capacity of the degraded SDH network and the steady state SDH network. On the y axis we have the ratio between the throughput obtained by the method in question, and the throughput of the steady state solution.

Fig 3. (a) shows results for the neural network combined with the heuristic second stage. Fig 3. (b) shows results for the recurrent neural network second stage. Fig 3. (c) shows results for the heuristic only, initialised by the steady state (Gopal-S) and Fig 3. (d) has the results for the heuristic initialised by zero (Gopal-0).

## 7 Discussion and Conclusions

The combined neural network/heuristic approach performs very well across the whole range of degrees of SDH network degradation tested. The results obtained in this paper are consistent with those found in [10]. The average accuracy of ~99% and fast solution generation times (< 60 ms) highlight this approach as a possible candidate for implementation in a real system, especially when one considers the easily achievable speed increase available from parallelizing the neural network. The mask perceptron used in these experiments is well suited for simulation on a DSP (or other hardware): the operations required are only comparisons, calculation of logical "AND" and the summation of synaptic weights (no multiplications or any non-linear transformations are required).

The interesting thing to note is the relatively good performance of the recurrent network, namely that it is able to handle over 80% of cases achieving very good performance when compared against the neural network/heuristic solution (95% of the quality of the teacher). One thing to bear in mind is that the heuristic approach is highly tuned to producing a solution which satisfies the constraints, changing the capacity of one link at a time until the desired goal is achieved. On the other hand the recurrent network is generic and does not target the constraints in such a specific manner, making quite crude global changes in

one hit, and yet is still able to achieve a reasonable level of performance. While the speed for the recurrent network was lower on average than for the heuristic solution in our experiments, this is not a major problem since many improvements are still possible and the results reported here are only preliminary, but serve to show what is possible. It is planned to continue the SDH network experiment in the future; with more investigation on the recurrent network for the second stage and also more complex SDH architectures.

## Acknowledgments

The research and development reported here has the active support of various sections and individuals within the Telstra Research Laboratories (TRL), especially Dr. C. Leckie and Mr. P. Sember who were responsible for the creation and trialling of the programs designed to produce the testing and training data.

The SDH application was possible due to co-operation of a number of our colleagues in TRL, in particular Dr. L. Campbell (who suggested this particular application), Dr. M. Herzberg and Mr. A. Herschtal.

The permission of the Managing Director, Research and Information Technology, Telstra, to publish this paper is acknowledged.

## References

[1]    E. Booker, Cross-connect at a Crossroads, Telephony, Vol. 215, 1988, pp. 63-65.

[2]    S. Bye, A Connectionist Approach to SDH Bandwidth Management, *Proceedings of the 19th International Conference on Artificial Neural Networks (ICANN-93)*, Brighton Conference Centre, UK, 1993, pp. 286-290.

[3]    R. Gillan, Advanced Network Architectures Exploiting the Synchronous Digital Hierarchy, Telecommunications Journal of Australia 39, 1989, pp. 39-42.

[4]    G. Gopal, C. Kim and A. Weinrib, Algorithms for Reconfigurable Networks, *Proceedings of the 13th International Teletraffic Congress (ITC-13)*, Copenhagen, Denmark, 1991, pp. 341-347.

[5]    M. Herzberg, Network Bandwidth Management - A New Direction in Network Management, *Proceedings of the 6th Australian Teletraffic Research Seminar*, Wollongong, Australia, pp. 218-225.

[6]    M. Herzberg and S. Bye, Bandwidth Management in Reconfigurable Networks, *Australian Telecommunications Research* 27, 1993, pp 57-70.

[7]    A. Kowalczyk and H.L. Ferra, Developing Higher Order Networks with Empirically Selected Units, *IEEE Transactions on Neural Networks*, pp. 698-711, 1994.

[8]    C. Leckie, A Connectionist Approach to Telecommunication Network Optimisation, in *Complex Systems: Mechanism of Adaptation*, R.J. Stonier and X.H. Yu, eds., IOS Press, Amsterdam, 1994.

[9]    M. Schwartz, *Telecommunications Networks*, Addison-Wesley, Readings, Massachusetts, 1987.

[10]   P. Campbell, H.L. Ferra, A. Kowalczyk, C. Leckie and P. Sember, Neural Networks in Real Time Decision Making, *Proceedings of the International Workshop on Applications of Neural Networks to Telecommunications 2 (IWANNT-95)*, Ed. J Alspector et. al. Lawrence Erlbaum Associates, New Jersey, 1995, pp. 273-280.